# A Domain Decomposition Method for Fast Manifold Learning

**Zhenyue Zhang**
Department of Mathematics
Zhejiang University, Yuquan Campus,
Hangzhou, 310027, P. R. China
zyzhang@zju.edu.cn

**Hongyuan Zha**
Department of Computer Science
Pennsylvania State University
University Park, PA 16802
zha@cse.psu.edu

## Abstract

We propose a fast manifold learning algorithm based on the methodology of domain decomposition. Starting with the set of sample points partitioned into two subdomains, we develop the solution of the interface problem that can glue the embeddings on the two subdomains into an embedding on the whole domain. We provide a detailed analysis to assess the errors produced by the gluing process using matrix perturbation theory. Numerical examples are given to illustrate the efficiency and effectiveness of the proposed methods.

## 1 Introduction

The setting of manifold learning we consider is the following. We are given a *parameterized manifold* of dimension $d$ defined by a mapping $f : \Omega \rightarrow \mathcal{R}^m$, where $d < m$, and $\Omega$ open and connected in $\mathcal{R}^d$. We assume the manifold is well-behaved, it is smooth and contains no self-intersections etc. Suppose we have a set of points $x_1, \cdots, x_N$, sampled possibly with noise from the manifold, i.e.,

$$x_i = f(\tau_i) + \epsilon_i, \quad i = 1, \ldots, N, \tag{1.1}$$

where $\epsilon_i$'s represent noise. The goal of manifold learning is to recover the parameters $\tau_i$'s and/or the mapping $f(\cdot)$ from the sample points $x_i$'s [2, 6, 9, 12]. The general framework of manifold learning methods involves imposing a connectivity structure such as a $k$-nearest-neighbor graph on the set of sample points and then turn the embedding problem into the solution of an eigenvalue problem. Usually constructing the graph dominates the computational cost of a manifold learning algorithm, but for large data sets, the computational cost of the eigenvalue problem can be substantial as well.

The focus of this paper is to explore the methodology of domain decomposition for developing fast algorithms for manifold learning. Domain decomposition by now is a well-established field in scientific computing and has been successfully applied in many science and engineering fields in connection with numerical solutions of partial differential equations. One class of domain decomposition methods partitions the solution domain into subdomains, solves the problem on each subdomain and glue the partial solutions on the subdomains by solving an interface problem [7, 10]. This is the general approach we will

follow in this paper. In particular, in section 3, we consider the case where the given set of sample points $x_1, \ldots, x_N$ are partitioned into two subdomains. On each of the subdomain, we can use a manifold learning method such as LLE [6], LTSA [12] or any other manifold learning methods to construct an embedding for the subdomain in question. We will then formulate the interface problem the solution of which will allow us to combine the embeddings on the two subdomains together to obtain an embedding over the whole domain. However, it is not always feasible to carry out the procedure described above. In section 2, we give necessary and sufficient conditions under which the embedding on the whole domain can be constructed from the embeddings on the subdomains. In section 4, we analyze the errors produced by the gluing process using matrix perturbation theory. In section 5, we briefly mention how the partitioning of the set of sample points into subdomains can be accomplished by some graph partitioning algorithms. Section 6 is devoted to numerical experiments.

NOTATION. We use $e$ to denote a column vector of all 1's the dimension of which should be clear from the context. $\mathcal{N}(\cdot)$ and $\mathcal{R}(\cdot)$ denote the null space and range space of a matrix, respectively. For an index set $I = [i_1, \ldots, i_k]$, $A(:, I)$ denotes the submatrix of $A$ consisting of columns of $A$ with indices in $I$ with a similar definition for the rows of a matrix. We use $\| \cdot \|$ to denote the spectral norm of a matrix.

## 2   A Basic Theorem

Let $X = [x_1, \cdots, x_N]$ with $x_i = f(\tau_i) + \epsilon_i, i = 1, \ldots, N$. Assume that the whole sample domain $X$ is divided into two subdomains $X_1 = \{x_i \,|\, i \in I_1\}$ and $X_2 = \{x_i \,|\, i \in I_2\}$. Here $I_1$ and $I_2$ denote the index sets such that $I_1 \cup I_2 = \{1, \ldots, N\}$ and $I_1 \cap I_2$ is not empty. Suppose we have obtained the two low-dimensional embeddings $T_1$ and $T_2$ of the sub-domains $X_1$ and $X_2$, respectively. The domain decomposition method attempts to recover the overall embedding $T = \{\tau_1, \ldots, \tau_N\}$ from the embeddings $T_1$ and $T_2$ on the subdomains.

In general, the recovered sub-embedding $T_j, j = 1, 2$, may not be exactly the subset $\{\tau_i \,|\, i \in I_j\}$ of $T$. For example, it is often the case that the recovered embeddings $T_j$ are approximately affinely equal to $\{\tau_i \,|\, i \in I_j\}$, i.e., up to certain approximation errors, there is an affine transformation such that

$$T_j = \{F_j \tau_i + c_j \,|\, i \in I_j\},$$

where $F_j$ is a nonsingular matrix and $c_j$ a column vector. Thus a domain decomposition method for manifold learning should be invariant to affine transformation on the embeddings $T_j$ obtained from subdomains. In that case, we can assume that $T_j$ is just the subset of $T$, i.e., $T_j = \{\tau_i \,|\, i \in I_j\}$. With an abuse of notation, we also denote by $T$ and $T_j$ the matrices of the column vectors in the set $T$ and $T_j$, for example, we write $T = [\tau_1, \ldots, \tau_N]$.

Let $\Phi_j$ be an orthogonal projection with $\mathcal{N}(\Phi_j) = \text{span}([e, T_j^T])$. Then $T_j$ can be recovered by computing the eigenvectors of $\Phi_j$ corresponding to its zero eigenvalues. To recover the whole $T$ we need to construct a matrix $\Phi$ with $\mathcal{N}(\Phi) = \text{span}([e, T^T])$ [11].

To this end, for each $T_j$, let $\Phi_j = Q_j Q_j^T \in \mathcal{R}^{N_j \times N_j}$, where $Q_j$ is an orthonormal basis matrix of $\mathcal{N}([e, T_j^T]^T)$ and $N_j$ is the column-size of $T_j$. To construct a $\Phi$ matrix, Let $S_j \in \mathcal{R}^{N \times N_j}$ be the 0-1 selection matrix defined as $S_j = I_N(:, I_j)$, where $I_N$ is the identity matrix of order $N$. Let $\hat{\Phi}_j = S_j \Phi_j S_j^T$. We then simply take $\Phi = \hat{\Phi}_1 + \hat{\Phi}_2$, or more flexibly, $\Phi = w_1 \hat{\Phi}_1 + w_2 \hat{\Phi}_2$, where $w_1$ and $w_2$ are the weights: $w_i > 0$ and $w_1 + w_2 = 1$. Obviously $\|\Phi\| \le 1$ since $\|\Phi_j\| = 1$. The following theorem gives the necessary and sufficient conditions under which the null space of $\Phi$ is just $\text{span}\{[e, T^T]\}$. (In the theorem, we only require the $\Phi_j$ to positive semidefinite.)

**Theorem 2.1** *Let $\Phi_i$ be two positive semidefinite matrices such that $\mathcal{N}(\Phi_i) = \mathrm{span}([e, T_i^T])$, $i = 1, 2$, and $T_0 = T_1 \cap T_2$. Assume that $[e, T_1^T]$ and $[e, T_2^T]$ are of full column-rank. Then $\mathcal{N}(\Phi) = \mathrm{span}([e, T^T])$ if and only if $[e, T_0^T]$ is of full column-rank.*

*Proof.* We first prove the necessity by contradiction. Assume that $\mathcal{N}([e, T_0^T]) \neq \mathcal{N}([e, T_2^T])$, then there is $y \neq 0$ such that $[e, T_0^T]y = 0$ and $[e, T^T(:, I_2)]y \neq 0$. Denote by $I_1^c$ the complement of $I_1$, i.e., the index set of $i$'s which do not belong to $I_1$. Then $[e, T^T(:, I_1^c)]y \neq 0$. Now we construct a vector $x$ as

$$x(I_1) = [e, T_1^T]y, \quad x(I_1^c) = 0.$$

Clearly $x(I_2) = 0$ and hence $x \in \mathcal{N}(\Phi)$. By the condition $\mathcal{N}(\Phi) = \mathrm{span}([e, T^T])$, we can write $x$ in the form $x = [e, T^T]z$ for a column vector $z$. Specially, $x(I_1) = [e, T_1^T]z$. Note that we also have $x(I_1) = [e, T_1^T]y$ by definition. It implies that $z = y$ because $[e, T_1^T]$ is of full rank. Therefor,

$$[e, T^T(:, I_1^c)]y = [e, T^T(:, I_1^c)]z = x(I_1^c) = 0.$$

Using it together with $[e, T_0^T]y = 0$ we have $[e, T^T(:, I_2)]y = 0$, a contradiction.

Now we prove the sufficiency. Let $Q$ be a basis matrix of $\mathcal{N}(\Phi)$. we have

$$w_1 G_1 Q^T \hat{\Phi}_1 Q + w_2 G_2 Q^T \hat{\Phi}_2 Q = Q^T \Phi Q = 0,$$

which implies $\Phi_i Q(I_1, :) = 0$, $i = 1, 2$, because $\hat{\Phi}_i$ is positive semidefinite. So

$$Q(I_i, :) = [e, T_i^T]G_i, \quad i = 1, 2. \tag{2.2}$$

Taking the overlap part $Q(I_0, :)$ of $Q$ with the different representations

$$Q(I_0, :) = [e, T_i(:, I_0)^T]G_i = [e, T_0^T]G_i,$$

we obtain $[e, T_0^T](G_1 - G_2) = 0$. So $G_1 = G_2$ because $[e, T_0^T]$ is of full column rank, giving rise to $Q = [e, T^T]G_1$, i.e., $\mathcal{N}(\Phi) \subset \mathrm{span}([e, T^T])$. It follows together with the obvious result $\mathrm{span}([e, T^T]) \subset \mathcal{N}(\Phi)$ that $\mathcal{N}(\Phi) = \mathrm{span}([e, T^T])$. ∎

The above result states that when the overlapping is large enough such that $[e, T_0^T]$ is of full column-rank (which is generically true when $T_0$ contains $d + 1$ points or more), the embedding over the whole domain can be recovered from the embeddings over the two subdomains. However, to follow Theorem 2.1, it seems that we will need to compute the null space of $\Phi$. In the next section, we will show this can done much cheaply by considering an interface problem which is of much smaller dimension.

## 3 Computing the Null Space of $\Phi$

In this section, we formulate the interface problem and show how to solve it to glue the embeddings from the two subdomains to obtain an embedding over the whole domain. To simplify notations, we re-denote by $T^*$ the *actual* embedding over the whole domain and $T_j^*$ the subsets of $T^*$ corresponding to subdomains. We then use $T_j$ to denote affinely transformed versions of $T_j^*$ obtained by LTSA for example, i.e., $T_j^* = c_j e^T + F_j T_j$. Here $c_j$ is a constant column vector in $\mathcal{R}^d$ and $F_j$ is a nonsingular matrix. Denote by $T_{0j}$ the overlapping part of $T_j$ corresponding to $I_0 = I_1 \cap I_2$ as in the proof of Theorem 2.1. We consider the overlapping parts $T_{0j}^*$ of $T_j^*$,

$$c_1 e^T + F_1 T_{01} = T_{01}^* = T_{02}^* = c_2 e^T + F_2 T_{02}. \tag{3.3}$$

Or equivalently,

$$\left[ [e, T_{01}^T], -[e, T_{02}^T] \right] \begin{bmatrix} (c_1, F_1)^T \\ (c_2, F_2)^T \end{bmatrix} = 0.$$

Therefore, if we take an orthonormal basis $G$ of the null space of $\left[[e, T_{01}^T], \ -[e, T_{02}^T]\right]$ and partition $G = [G_1^T, G_2^T]^T$ conformally, then $[e, T_{01}^T]G_1 = [e, T_{02}^T]G_2$. Let $A_j = G_j^T[e, T_j^T]^T$, $j = 1, 2$. Define the matrix $A$ such that $A(:, I_j) = A_j$. Then since $\Phi_i A_i^T = 0$, the well-defined matrix $A^T$ is a basis of $\mathcal{N}(\Phi)$,

$$\Phi A^T = S_1 \Phi_1 S_1^T A^T + S_2 \Phi_2 S_2^T A^T = S_1 \Phi_1 A_1^T + S_2 \Phi_2 A_2^T = 0.$$

Therefore, we can use $A^T$ to recover the global embedding $T$.

A simpler alternative way is use a one-sided affine transformation, i.e., fix one of $T_i$ and affinely transform the other; the affine matrix is obtained by fixing one of $\tilde{T}_{0i}$ and transforming the other. For example, we can determine $c$ and $F$ such that

$$T_{01} = ce^T + FT_{02}, \tag{3.4}$$

and transform $T_2$ to $\hat{T}_2 = ce^T + FT_2$. Clearly, for the overlapping part, $\hat{T}_{02} = T_{01}$. Then we can construct a larger matrix $T$ by $T(:, I_1) = T_1$, $T(:, I_2) = ce^T + FT_2$. One can also readily verify that $T^T$ is a basis matrix of $\mathcal{N}(\Phi)$.

In the noisy case, a least squares formulation will be needed. For example, for the simultaneous affine transformation, we take $G = [G_1^T, G_2^T]^T$ to be an orthonormal matrix in $\mathcal{R}^{2(d+1)\times(d+1)}$ such that

$$\|[e, T_{01}^T]G_1 - [e, T_{02}^T]G_2\| = \min.$$

It is known that the minimum $G$ is given by the right singular vector matrix corresponding to the $d + 1$ smallest singular values of $W = \left[[e, T_{01}^T], \ -[e, T_{02}^T]\right]$, and the residual $\left\|[e, T_{01}^T]G_1 - [e, T_{02}^T]G_2\right\| = \sigma_{d+2}(W)$. For the one-side approach (3.4), $[c, F]$ can be a solution to the least squares problem

$$\min_{c,\, F} \left\|T_{01} - \left(ce^T + FT_{02}\right)\right\| = \min_{F} \left\|(T_{01} - t_{01}e^T) - F(T_{02} - t_{02}e^T)\right\|,$$

where $t_{0j}$ is the column mean of $T_{0j}$. The minimum is achieved at $F = (T_{01} - t_{01}e^T)(T_{02} - t_{02}e^T)^+$, $c = t_{01} - Ft_{02}$. Clearly, the residual now reads as

$$\min_{c,\, F} \left\|T_{01} - \left(ce^T + FT_{02}\right)\right\| = \left\|(T_{01} - t_{01}e^T)\left(I - (T_{02} - t_{02}e^T)^+(T_{02} - t_{02}e^T)\right)\right\|.$$

Notice that the overlapping parts in the two affinely transformed subsets are not exactly equal to each other in the noisy case. There are several possible choices for setting $A(:, I_0)$ or $\hat{T}(:, I_0)$. For example, one choice is to set $T(:, I_0)$ by a convex combination of $T_{0j}$'s,

$$T(:, I_0) = \alpha T_{01} + (1 - \alpha)\hat{T}_{02}.$$

with $\alpha = 1/2$ for example.

We summarize discussions above in the following two algorithms for gluing the two sub-domains $T_1$ and $T_2$.

**Algorithm** I. [Simultaneously affine transformation]

1. Compute the right singular vector matrix $G$ corresponding to the $d + 1$ smallest singular values of $\left[[e, T_{01}^T], \ -[e, T_{02}^T]\right]$.

2. Partition $G = [G_1^T, G_2^T]^T$ and set $A_i = G_i^T[e, T_i^T]^T$, $i = 1, 2$, and

$$A(:, I_1 \backslash I_0) = A_{11}, \quad A(:, I_0) = \alpha A_{01} + (1 - \alpha)A_{02}, \quad A(:, I_2 \backslash I_0) = A_{12},$$

where $A_{0j}$ is the overlap part of $A_j$ and $A_{1j}$ is the $A_j$ with $A_{0j}$ deleted.

3 Compute the column mean $a$ of $A$, and an orthogonal basis $U$ of $\mathcal{N}(a^T)$.

4. Set $T = U^T A$.

**Algorithm** II. [One-side affine transformation]

1. Compute the least squares problem $\min_W \|T_{01} - W[e, T_{02}^T]^T\|_F$.

2. Affinely transform $T_2$ to $\hat{T}_2 = W[e, T_2^T]^T$.

3. Set the global coordinate matrix $T$ by

$$T(:, I_1 \backslash I_0) = T_{11}, \quad T(:, I_0) = \alpha T_{01} + (1 - \alpha)\hat{T}_{02}, \quad T(:, I_2 \backslash I_0) = \hat{T}_{12}.$$

## 4  Error Analysis

As we mentioned before, the computation of $T_j, j = 1, 2$ using a manifold learning algorithm such as LTSA involves errors. In this section, we assess the impact of those errors on the accuracy of the gluing process. Two issues are considered for the error analysis. One is the perturbation analysis of $\mathcal{N}(\Phi^*)$ when the computation of $\Phi_i^*$ is subject to error. In this case, $\mathcal{N}(\Phi^*)$ will be approximated by the smallest $(d+1)$-dimensional eigenspace $\mathcal{V}$ of an approximation $\Phi \approx \Phi^*$ (Theorem 4.1). The other issue is the error estimation of $\mathcal{V}$ when a basis matrix of $\mathcal{V}$ is approximately constructed by affinely transformed local embeddings as described in section 3 (Theorem 4.2). Because of space limit, we will not present the details of the proofs of the results.

The distance of two linear subspaces $\mathcal{X}$ and $\mathcal{Y}$ are defined by $\text{dist}(\mathcal{X}, \mathcal{Y}) = \|P_{\mathcal{X}} - P_{\mathcal{Y}}\|$, where $P_{\mathcal{X}}$ and $P_{\mathcal{Y}}$ are the orthogonal projection onto $\mathcal{X}$ and $\mathcal{Y}$, respectively. Let $\epsilon_i = \|\Phi_i - \Phi_i^*\|$, where $\Phi_i^*$ and $\Phi_i$ are the orthogonal projectors onto the range spaces $\text{span}([e, (T_i^*)^T])$ and $\text{span}([e, (T_i)^T])$, respectively. Clearly, if $\Phi^* = w_1 \Phi_1^* + w_2 \Phi_2^*$ and $\Phi = w_1 \Phi_1 + w_2 \Phi_2$, then

$$\text{dist}\Big(\text{span}([e, (T^*)^T]), \text{span}([e, T^T])\Big) = \|\Phi - \Phi^*\| \le w_1 \epsilon_1 + w_2 \epsilon_2 \equiv \epsilon.$$

**Theorem 4.1** *Let $\sigma$ be the smallest nonzero eigenvalue of $\Phi^*$ and $\mathcal{V}$ the subspace spanned by the eigenvectors of $\Phi$ corresponding to the $d + 1$ smallest eigenvalues. If $\epsilon < \sigma/4$, and $4\epsilon^2(\|\Phi^*\| - \sigma + 2\epsilon) < (\sigma - 2\epsilon)^3$, then*

$$\text{dist}(\mathcal{V}, \mathcal{N}(\Phi^*)) \le \frac{\epsilon}{\sqrt{(\sigma/2 - \epsilon)^2 + \epsilon^2}}.$$

**Theorem 4.2** *Let $\sigma$ and $\epsilon$ be defined in Theorem 4.1. A is the matrix computed by the simultaneous affine transformation (Algorithm I in section 3) Let $\sigma_i(\cdot)$ be the $i$-th smallest singular value of a matrix. Denote*

$$\mu = \frac{1}{2}\sigma_{d+2}([[e, T_{01}^T], -[e, T_{02}^T]]), \quad \eta = \frac{\mu}{\sigma_{\min}(A)}.$$

*If $\epsilon < \sigma/4$, then*

$$\text{dist}(\mathcal{V}, \text{span}(A)) \le \frac{1}{\sigma_{d+2}(\Phi)}\Big(\eta + \frac{\epsilon\sigma/2}{(\sigma/2 - \epsilon)^2}\Big)$$

From Theorems 4.1 and 4.2 we conclude directly that

$$\text{dist}(\text{span}(A), \mathcal{N}(\Phi^*)) \le \frac{1}{\sigma_{d+2}(\Phi)}\Big(\eta + \frac{\epsilon\sigma/2}{(\sigma/2 - \epsilon)^2}\Big) + \frac{2\epsilon}{\sqrt{(\sigma - 2\epsilon)^2 + 4\epsilon^2}}.$$

# 5  Partitioning the Domains

To apply the domain decomposition methods, we need to partition the given set of data points into several domains making use of the $k$ nearest neighbor graph imposed on the data points. This reduces the problem to a graph partition problem and many techniques such as spectral graph partitioning and METIS [3, 5] can be used. In our experiments, we have used a particularly simple approach: we use the reverse Cuthill-McKee method [4] to order the vertices of the $k$-NN graph and then partition the vertices into domains (for details see Test 2 in the next section).

Once we have partitioned the whole domain into multiple overlapping subdomains we can use the following two approaches to glue them together.

**Successive gluing.** Here we glue the subdomains one by one as follows. Initially set $T^{(1)} = T_1$ and $I^{(1)} = I_1$, and then glue the patch $T_k$ to $T^{(k-1)}$ and obtain the larger one $T^{(k)}$ for $k = 2, \ldots, K$, and so on. The index set of $T^{(k)}$ is given by $I^{(k)} = I^{(k-1)} \cup I_k$. Clearly the overlapping set of $T^{(k-1)}$ and $T_k$ is $I_0^{(k)} = I^{(k-1)} \cap I_k$.

**Recursive gluing.** Here at the leaf level, we divide the subdomains into several pairs, say $(T_{2i-1}^{(0)}, T_{2i}^{(0)})$, $1 = 1, 2, \ldots$. Then glue each pair to be a larger subdomain $T_i^{(1)}$ and continue. The recursive gluing method is obviously parallelizable.

# 6  Numerical Experiments

In this section we report numerical experiments for the proposed domain decomposition methods for manifold learning. This efficiency and effectiveness of the methods clearly depend on the accuracy of the computed embeddings for subdomains, the sizes of the subdomains, and the sizes of the overlaps of the subdomains.

**Test 1.** Our first test data set is sampled from a Swiss-roll as follows

$$x_i = [t_i \cos(t_i), h_i, t_i \sin(t_i)]^T, \quad i = 1, \ldots, N = 2000, \tag{6.5}$$

where $t_i$ and $h_i$ are uniformly randomly chosen in the intervals $[\frac{3\pi}{2}, \frac{9\pi}{2}]$ and $[0, 21]$, respectively. Let $\tau_i$ be the arc length of the corresponding spiral curve $[t \cos(t), t \sin(t)]^T$ from $t_0 = \frac{3\pi}{2}$ to $t_i$. $\tau_{\max} = \max_i \tau_i$. To compare the CPU time of the domain decomposition methods, we simply partition the $\tau$-interval $[0, \tau_{\max}]$ into $k_\tau$ subintervals $(a_{i-1}, a_i]$ with equal length and also partition the $h$-interval into $k_h$ subintervals $(b_{j-1}, b_j]$. Let $D_{ij} = (a_{i-1}, a_i] \times (b_{j-1}, b_j]$ and $S_{ij}(r)$ be the balls centered at $(a_i, b_j)$ with radius $r$. We set the subdomains as

$$X_{ij} = \{x_k \,|\, (\tau_k, h_k) \in D_{ij} \cup S_{ij}(r)\}.$$

Clearly $r$ determines the size of overlapping parts of $X_{ij}$ with $X_{i+1,j}, X_{i,j+1}, X_{i+1,j+1}$. The submatrices $X_{ij}$ are ordered as $X_{1,1}, X_{1,2}, \ldots, X_{1,k_h}, X_{2,1}, \ldots$ and denoted as $X_k$, $k = 1, \ldots, K = k_\tau k_h$. We first compute the $K$ local 2-D embeddings $T_1, \ldots, T_K$ by applying LTSA on the sample data sets $X_k$ for the subdomains. Then those local coordinate embeddings $T_k$ are aligned by the successive one-sided affine transformation algorithm by adding subdomain $T_k$ one by one.

Table 1 lists the total CPU time for the successive domain decomposition algorithm, including the time for computing the embeddings $\{T_k\}$ for the subdomains, for different parameters $k_\tau$ and $k_h$ with the parameter $r = 5$. In Table 2, we list the CPU time for the recursive gluing approach taking into account the parallel procedure. As a comparison, the CPU time of LTSA applying to the whole data points is $6.23$ seconds.

Table 1: CPU Time (seconds) of the successive domain decomposition algorithm.

|            | $k_h=2$ | 3    | 4    | 5    | 6    |
|------------|---------|------|------|------|------|
| $k_\tau=3$ | 1.89    | 1.70 | 1.64 | 1.61 | 1.64 |
| 4          | 167     | 1.67 | 1.61 | 1.70 | 1.77 |
| 5          | 1.66    | 1.59 | 1.67 | 1.78 | 1.86 |
| 6          | 163     | 1.66 | 1.75 | 1.89 | 2.09 |
| 7          | 1.59    | 1.70 | 1.84 | 2.02 | 2.23 |
| 8          | 1.58    | 1.80 | 1.94 | 2.22 | 2.44 |
| 9          | 1.63    | 1.83 | 2.06 | 2.31 | 2.66 |
| 10         | 1.63    | 1.86 | 2.38 | 2.56 | 2.94 |

Table 2: CPU Time (seconds) of the parallel recursive domain decomposition.

|            | $k_h=2$ | 3    | 4    | 5    | 6    |
|------------|---------|------|------|------|------|
| $k_\tau=3$ | 0.52    | 0.34 | 0.27 | 0.19 | 017  |
| 4          | 0.53    | 0.23 | 0.20 | 0.17 | 0.13 |
| 5          | 0.31    | 0.17 | 0.19 | 0.17 | 0.14 |
| 6          | 0.25    | 0.19 | 0.16 | 0.13 | 0.14 |
| 7          | 0.20    | 0.16 | 0.14 | 0.14 | 0.11 |
| 8          | 0.20    | 0.17 | 0.16 | 0.14 | 0.14 |
| 9          | 0.19    | 0.16 | 0.14 | 0.14 | 0.14 |
| 10         | 0.19    | 0.16 | 0.17 | 0.19 | 0.13 |

**Test 2.** The symmetric reverse Cuthill-McKee permutation (*symrcm*) is an algorithm for ordering the rows and columns of a symmetric sparse matrix [4]. It tends to move the nonzero elements of the sparse matrix towards the main diagonals of the matrix. We use Matlab's *symrcm* to the adjacency matrix of the k-nearest-neighbor graph of the data points to reorder them. Denote by $X$ the reordered data set. We then partition the whole sample points into $K = 16$ subsets $X_i = X(:, s_i : e_i)$ with $s_i = \max\{1, (i-1)m - 20\}$, $e_i = \min\{im + 20, N\}$, and $m = N/K = 125$.

It is known that the $t$-$h$ parameters in (6.5) represent an isometric parametrization of the swiss-roll surface. We have shown that within the errors made in computing the local embeddings, LTSA can recover the isometric parametrization up to an affine transformation [11]. We denote by $\tilde{T}^{(k)} = ce^T + FT^{(k)}$ the optimal approximation to $T^*(:, I^{(k)})$ within affine transformations,

$$\|T^*(:, I^{(k)}) - \tilde{T}^{(k)}\|_F = \min_{c,F} \|T^*(:, I^{(k)}) - (ce^T + FT^{(k)})\|_F.$$

We denote by $\eta_k$ the average of relative errors

$$\eta_k = \frac{1}{|I^{(k)}|} \sum_{i \in I^{(k)}} \frac{\|T^*(:, i) - \tilde{T}^{(k)}(:, i)\|_2}{\|T^*(:, i)\|_2}.$$

In the left panel of Figure 1 we plot the initial embedding errors for the subdomains (blue bar), the error of LTSA applied to the whole data set (red bar), and the errors $\eta_k$ of the successive gluing (red line). The successive gluing method gives an embedding with an acceptable accuracy comparing with the accuracy obtained by applying LTSA to the whole data set. As shown in the error analysis, the errors in successive gluing will increase when the initial errors for the subdomains increase. To show it more clearly, we also plot the $\eta_k$ for the recursive gluing method in the right panel of Figure 1.

**Acknowledgment.** The work of first author was supported in part by by NSFC (project 60372033), the Special Funds for Major State Basic Research Projects (project

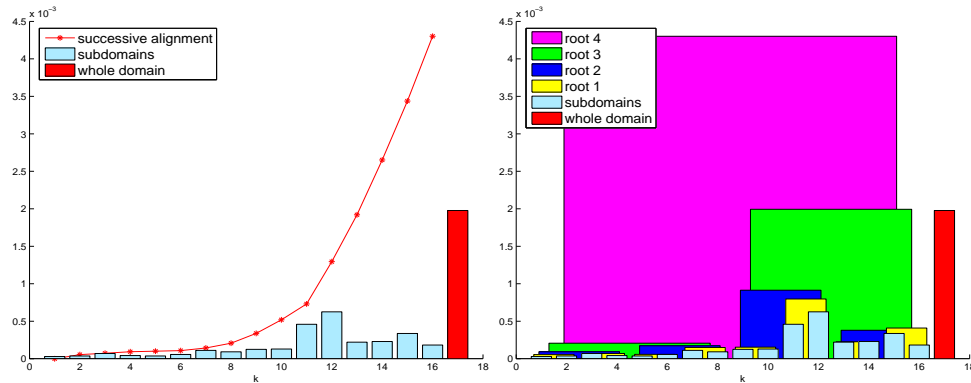

Figure 1: Relative errors for the successive (left) and recursive (right) approaches.

G19990328), and NSF grant CCF-0305879. The work of second author was supported in part by NSF grants DMS-0311800 and CCF-0430349.

## References

[1] M. Brand. Charting a manifold. *Advances in Neural Information Processing Systems* 15, MIT Press, 2003.

[2] D. Donoho and C. Grimes. Hessian Eigenmaps: new tools for nonlinear dimensionality reduction. *Proceedings of National Academy of Science*, 5591-5596, 2003.

[3] M. Fiedler. A property of eigenvectors of nonnegative symmetric matrices and its application to graph theory. *Czech. Math. J.* 25:619–637, 1975.

[4] A. George and J. W. Liu. *Computer Solution of Large Sparse Positive Definite Matrices*. Prentice Hall, 1981.

[5] METIS. `http://www-users.cs.umn.edu/~karypis/metis/`.

[6] S. Roweis and L. Saul. Nonlinear dimensionality reduction by locally linear embedding. *Science*, 290: 2323–2326, 2000.

[7] B. Smith, P. Bjorstad and W. Gropp *Domain Decomposition, Parallel Multilevel Methods for Elliptic Partial Differential Equations*. Cambridge University Press, 1996.

[8] G.W. Stewart and J.G. Sun. *Matrix Perturbation Theory*. Academic Press, New York, 1990.

[9] J. Tenenbaum, V. De Silva and J. Langford. A global geometric framework for nonlinear dimension reduction. *Science*, 290:2319–2323, 2000.

[10] A. Toselli and O. Widlund. *Domain Decomposition Methods - Algorithms and Theory*. Springer, 2004.

[11] H. Zha and Z. Zhang. Spectral analysis of alignment in manifold learning. Proceedings of *IEEE International Conference on Acoustics, Speech, and Signal Processing,* (ICASSP), 2005.

[12] Z. Zhang and H. Zha. Principal manifolds and nonlinear dimensionality reduction via tangent space alignment. *SIAM J. Scientific Computing.* 26:313-338, 2005.